# Churn Reduction in the Wireless Industry

**Michael C. Mozer*+, Richard Wolniewicz*, David B. Grimes*+,**
**Eric Johnson*, Howard Kaushansky***

| * Athene Software | + Department of Computer Science |
|---|---|
| 2060 Broadway, Suite 300 | University of Colorado |
| Boulder, CO 80302 | Boulder, CO 80309–0430 |

## Abstract

Competition in the wireless telecommunications industry is rampant. To maintain profitability, wireless carriers must control *churn*, the loss of subscribers who switch from one carrier to another. We explore statistical techniques for churn prediction and, based on these predictions, an optimal policy for identifying customers to whom incentives should be offered to increase retention. Our experiments are based on a data base of nearly 47,000 U.S. domestic subscribers, and includes information about their usage, billing, credit, application, and complaint history. We show that under a wide variety of assumptions concerning the cost of intervention and the retention rate resulting from intervention, churn prediction and remediation can yield significant savings to a carrier. We also show the importance of a data representation crafted by domain experts.

Competition in the wireless telecommunications industry is rampant. As many as seven competing carriers operate in each market. The industry is extremely dynamic, with new services, technologies, and carriers constantly altering the landscape. Carriers announce new rates and incentives weekly, hoping to entice new subscribers and to lure subscribers away from the competition. The extent of rivalry is reflected in the deluge of advertisements for wireless service in the daily newspaper and other mass media.

The United States had 69 million wireless subscribers in 1998, roughly 25% of the population. Some markets are further developed; for example, the subscription rate in Finland is 53%. Industry forecasts are for a U.S. penetration rate of 48% by 2003. Although there is significant room for growth in most markets, the industry growth rate is declining and competition is rising. Consequently, it has become crucial for wireless carriers to control *churn*—the loss of customers who switch from one carrier to another. At present, domestic monthly churn rates are 2-3% of the customer base. At an average cost of $400 to acquire a subscriber, churn cost the industry nearly $6.3 billion in 1998; the total annual loss rose to nearly $9.6 billion when lost monthly revenue from subscriber cancellations is considered (Luna, 1998). It costs roughly five times as much to sign on a new subscriber as to retain an existing one. Consequently, for a carrier with 1.5 million subscribers, reducing the monthly churn rate from 2% to 1% would yield an increase in annual earnings of at least $54 million, and an increase in shareholder value of approximately $150 million. (Estimates are even higher when lost monthly revenue is considered; see Fowlkes, Madan, Andrew, & Jensen, 1999; Luna, 1998.)

The goal of our research is to evaluate the benefits of predicting churn using techniques from statistical machine learning. We designed models that predict the probability

of a subscriber churning within a short time window, and we evaluated how well these predictions could be used for decision making by estimating potential cost savings to the wireless carrier under a variety of assumptions concerning subscriber behavior.

# 1 THE FRAMEWORK

Figure 1 shows a framework for churn prediction and profitability maximization. Data from a subscriber—on which we elaborate in the next section—is fed into three components which estimate: the likelihood that the subscriber will churn, the profitability (expected monthly revenue) of the subscriber, and the subscriber's credit risk. Profitability and credit risk determine how valuable the subscriber is to the carrier, and hence influences how much the carrier should be willing to spend to retain the subscriber. Based on the predictions of subscriber behavior, a decision making component determines an *intervention strategy*—whether a subscriber should be contacted, and if so, what incentives should be offered to appease them. We adopt a decision-theoretic approach which aims to maximize the expected profit to the carrier.

In the present work, we focus on churn prediction and utilize simple measures of subscriber profitability and credit risk. However, current modeling efforts are directed at more intelligent models of profitability and credit risk.

# 2 DATA SET

The subscriber data used for our experiments was provided by a major wireless carrier. The carrier does not want to be identified, as churn rates are confidential. The carrier provided a data base of 46,744 primarily business subscribers, all of whom had multiple services. (Each service corresponds to a cellular telephone or to some other service, such as voice messaging or beeper capability.) All subscribers were from the same region of the United States, about 20% in major metropolitan areas and 80% more geographically distributed. The total revenue for all subscribers in the data base was $14 million in October 1998. The average revenue per subscriber was $234. We focused on multi-service subscribers, because they provide significantly more revenue than do typical single-service subscribers.

When subscribers are on extended contracts, churn prediction is relatively easy: it seldom occurs during the contract period, and often occurs when the contract comes to an end. Consequently, all subscribers in our data base were month-to-month, requiring the use of more subtle features than contract termination date to anticipate churn.

The subscriber data was extracted from the time interval October through December, 1998. Based on these data, the task was to predict whether a subscriber would churn in January *or* February 1999. The carrier provided their internal definition of churn, which was based on the closing of all services held by a subscriber. From this definition, 2,876 of the subscribers active in October through December churned—6.2% of the data base.

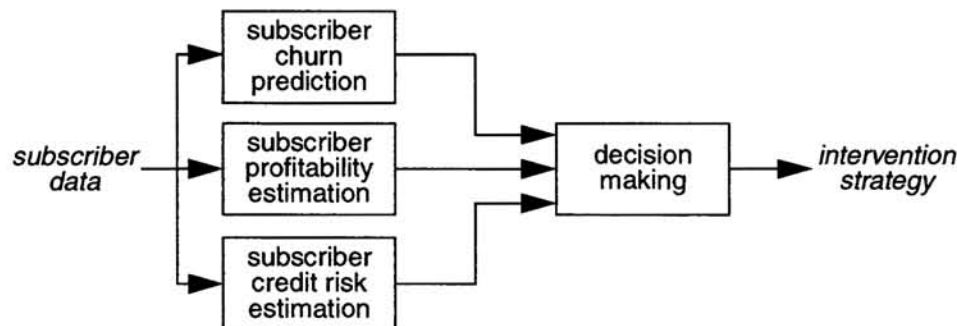

**FIGURE 1. The framework for churn prediction and profitability maximization**

## 2.1 INPUT FEATURES

Ultimately, churn occurs because subscribers are dissatisfied with the price or quality of service, usually as compared to a competing carrier. The main reasons for subscriber dissatisfaction vary by region and over time. Table 1 lists important factors that influence subscriber satisfaction, as well as the relative importance of the factors (J. D. Power and Associates, 1998). In the third column, we list the type of information required for determining whether a particular factor is likely to be influencing a subscriber. We categorize the types of information as follows.

**Network**. Call detail records (date, time, duration, and location of all calls), dropped calls (calls lost due to lack of coverage or available bandwidth), and quality of service data (interference, poor coverage).

**Billing**. Financial information appearing on a subscriber's bill (monthly fee, additional charges for roaming and additional minutes beyond monthly prepaid limit).

**Customer Service**. Calls to the customer service department and their resolutions.

**Application for Service**. Information from the initial application for service, including contract details, rate plan, handset type, and credit report.

**Market**. Details of rate plans offered by carrier and its competitors, recent entry of competitors into market, advertising campaigns, etc.

**Demographics**. Geographic and population data of a given region.

A subset of these information sources were used in the present study. Most notably, we did not utilize market information, because the study was conducted over a fairly short time interval during which the market did not change significantly. More important, the market forces were fairly uniform in the various geographic regions from which our subscribers were selected. Also, we were unable to obtain information about the subscriber equipment (age and type of handset used).

The information sources listed above were distributed over three distinct data bases maintained by the carrier. The data bases contained thousands of fields, from which we identified 134 variables associated with each subscriber which we conjectured might be linked to churn. The variables included: subscriber location, credit classification, customer classification (e.g., corporate versus retail), number of active services of various types, beginning and termination dates of various services, avenue through which services were activated, monthly charges and usage, number, dates and nature of customer service calls, number of calls made, and number of abnormally terminated calls.

## 2.2 DATA REPRESENTATION

As all statisticians and artificial intelligence researchers appreciate, representation is key. A significant portion of our effort involved working with domain experts in the wireless telecommunications industry to develop a representation of the data that highlights and makes explicit those features which—in the expert's judgement—were highly related to churn. To evaluate the benefit of carefully constructing the representation, we performed

**TABLE 1. Factors influencing subscriber satisfaction**

| Factor | Importance | Nature of data required for prediction |
|---|---|---|
| call quality | 21% | network |
| pricing options | 18% | market, billing |
| corporate capability | 17% | market, customer service |
| customer service | 17% | customer service |
| credibility / customer communications | 10% | market, customer service |
| roaming / coverage | 7% | network |
| handset | 4% | application |
| billing | 3% | billing |
| cost of roaming | 3% | market, billing |

studies using both *naive* and a *sophisticated* representations.

The naive representation mapped the 134 variables to a vector of 148 elements in a straightforward manner. Numerical variables, such as the length of time a subscriber had been with the carrier, were translated to an element of the representational vector which was linearly related to the variable value. We imposed lower and upper limits on the variables, so as to suppress irrelevant variation and so as not to mask relevant variation by too large a dynamic range; vector elements were restricted to lie between −4 and +4 standard deviations of the variable. One-of-$n$ discrete variables, such as credit classification, were translated into an $n$-dimensional subvector with one nonzero element.

The sophisticated representation incorporated the domain knowledge of our experts to produce a 73-element vector encoding attributes of the subscriber. This representation collapsed across some of the variables which, in the judgement of the experts, could be lumped together (e.g., different types of calls to the customer service department), and expanded on others (e.g., translating the scalar length-of-time-with-carrier to a multidimensional basis-function representation, where the receptive-field centers of the basis functions were suggested by the domain experts), and performed transformations of other variables (e.g., ratios of two variables, or time-series regression parameters).

## 3  PREDICTORS

The task is to predict the probability of churn from the vector encoding attributes of the subscriber. We compared the churn-prediction performance of two classes of models: logit regression and a nonlinear neural network with a single hidden layer and weight decay (Bishop, 1995). The neural network model class was parameterized by the number of units in the hidden layer and the weight decay coefficient. We originally anticipated that we would require some model selection procedure, but it turned out that the results were remarkably insensitive to the choice of the two neural network parameters; weight decay up to a point seemed to have little effect, and beyond that point it was harmful, and varying the number of hidden units from 5 to 40 yielded nearly identical performance. We likely were not in a situation where overfitting was an issue, due to the large quantity of data available; hence increasing the model complexity (either by increasing the number of hidden units or decreasing weight decay) had little cost.

Rather than selecting a single neural network model, we averaged the predictions of an ensemble of models which varied in the two model parameters. The average was uniformly weighted.

## 4  METHODOLOGY

We constructed four predictors by combining each of the two model classes (logit regression and neural network) with each of the two subscriber representations (naive and sophisticated). For each predictor, we performed a ten-fold cross validation study, utilizing the same splits across predictors. In each split of the data, the ratio of churn to no churn examples in the training and validation sets was the same as in the overall data set.

For the neural net models, the input variables were centered by subtracting the means and scaled by dividing by their standard deviation. Input values were restricted to lie in the range [−4, +4]. Networks were trained until they reached a local minimum in error.

## 5  RESULTS AND DISCUSSION

### 5.1  CHURN PREDICTION

For each of the four predictors, we obtain a predicted probability of churn for each subscriber in the data set by merging the test sets from the ten data splits. Because decision making ultimately requires a "churn" or "no churn" prediction, the continuous probability measure must be thresholded to obtain a discrete predicted outcome.

For a given threshold, we determine the proportion of churners who are correctly identified as churners (the *hit* rate), and the proportion of nonchurners who are correctly identified as nonchurners (the *rejection* rate). Plotting the hit rate against the rejection rate for various thresholds, we obtain an *ROC curve* (Green & Swets, 1966). In Figure 2, the closer a curve comes to the upper right corner of the graph—100% correct prediction of churn and 100% correct prediction of nonchurn—the better is the predictor at discriminating churn from nonchurn. The dotted diagonal line indicates no discriminability: If a predictor randomly classifies $x\%$ of cases as churn, it is expected to obtain a hit rate of $x\%$ and a rejection rate of $(100–x)\%$.

As the Figure indicates, discriminability is clearly higher for the sophisticated representation than for the naive representation. Further, for the sophisticated representation at least, the nonlinear neural net outperforms the logit regression. It appears that the neural net can better exploit nonlinear structure in the sophisticated representation than in the naive representation, perhaps due to the basis-function representation of key variables. Although the four predictors appear to yield similar curves, they produce large differences in estimated cost savings. We describe how we estimate cost savings next.

## 5.2 DECISION MAKING

Based on a subscriber's predicted churn probability, we must decide whether to offer the subscriber some *incentive* to remain with the carrier, which will presumably reduce the likelihood of churn. The incentive will be offered to any subscriber whose churn probability is above a certain threshold. The threshold will be selected to maximize the expected cost savings to the carrier; we will refer to this as the *optimal decision-making policy*.

The cost savings will depend not only on the discriminative ability of the predictor, but also on: the cost to the carrier of providing the incentive, denoted $C_i$ (the cost to the carrier may be much lower than the value to the subscriber, e.g., when air time is offered); the time horizon over which the incentive has an effect on the subscriber's behavior; the reduction in probability that the subscriber will leave within the time horizon as a result of the incentive, $P_i$; and the lost-revenue cost that results when a subscriber churns, $C_l$.

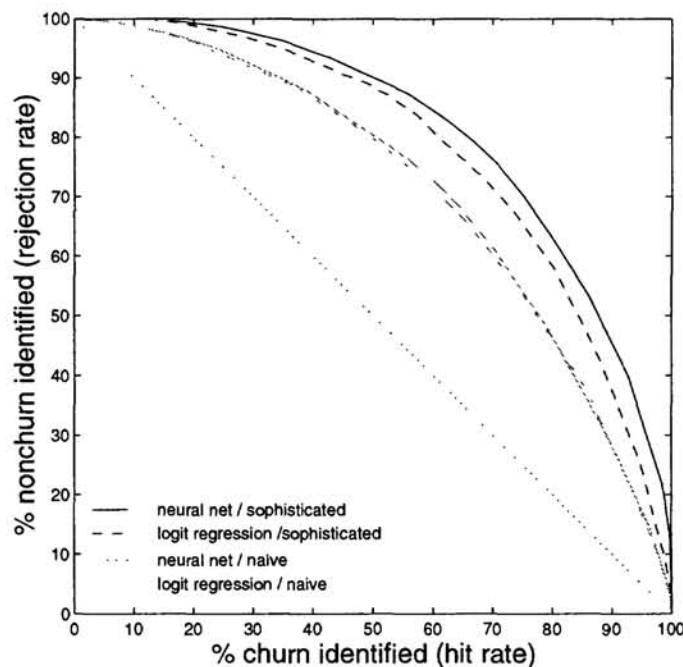

FIGURE 2. Test-set performance for the four predictors. Each curve shows, for various thresholds, the ability of a predictor to correctly identify churn (x axis) and nonchurn (y axis). The more bowed a curve, the better able a predictor is at discriminating churn from nonchurn.

We assume a time horizon of six months. We also assume that the lost revenue as a result of churn is the average subscriber bill over the time horizon, along with a fixed cost of $500 to acquire a replacement subscriber. (This acquisition cost is higher than the typical cost we stated earlier because subscribers in this data base are high valued, and often must be replaced with multiple low-value subscribers to achieve the same revenue.) To estimate cost savings, the parameters $C_i$, $P_i$, and $C_l$ are combined with four statistics obtained from a predictor:

$N(pL,aL)$:    number of subscribers who are predicted to leave (churn) and who actually leave barring intervention

$N(pS,aL)$:    number of subscribers who are predicted to stay (nonchurn) and who actually leave barring intervention

$N(pL,aS)$:    number of subscribers who are predicted to leave and who actually stay

$N(pS,aS)$:    number of subscribers who are predicted to stay and who actually stay

Given these statistics, the net cost to the carrier of performing no intervention is:

$$net(no\ intervention) = [\ N(pL,aL) + N(pS,aL)\ ]\ C_l$$

This equation says that whether or not churn is predicted, the subscriber will leave, and the cost per subscriber will be $C_l$. The net cost of providing an incentive to all subscribers whom are predicted to churn can also be estimated:

$$net(incentive) = [\ N(pL,aL) + N(pL,aS)\ ]\ C_i + [\ P_i\ N(pL,aL) + N(pS,aL)\ ]\ C_l$$

This equation says that the cost of offering the incentive, $C_i$, is incurred for all subscribers for who are predicted to churn, but the lost revenue cost will decrease by a fraction $P_i$ for the subscribers who are correctly predicted to churn. The savings to the carrier as a result of offering incentives based on the churn predictor is then

$$savings\ per\ churnable\ subscriber =$$
$$[\ net(no\ intervention) - net(incentive)\ ]\ /\ [\ N(pL,aL) + N(pS,aL)\ ]$$

The contour plots in Figure 3 show expected savings per churnable subscriber, for a range of values of $C_i$, $P_i$, and $C_l$, based on the optimal policy and the sophisticated neural-net predictor. Each plot assumes a different subscriber retention rate (= $1-P_i$) given intervention. The "25% retention rate" graph supposes that 25% of the churning subscribers who are offered an incentive will decide to remain with the carrier over the time horizon of six months. For each plot, the cost of intervention ($C_i$) is varied along the x-axis, and the average monthly bill is varied along the y-axis. (The average monthly bill is converted to lost revenue, $C_l$, by computing the total bill within the time horizon and adding the subscriber acquisition cost.) The shading of a region in the plot indicates the expected savings assuming the specified retention rate is achieved by offering the incentive. The grey-level bar to the right of each plot translates the shading into dollar savings per subscriber who will churn barring intervention. Because the cost of the incentive is factored into the savings estimate, the estimate is actually the net return to the carrier.

The white region in the lower right portion of each graph is the region in which no cost savings will be obtained. As the graphs clearly show, if the cost of the incentive needed to achieve a certain retention rate is low and the cost of lost revenue is high, significant per-subscriber savings can be obtained.

As one might suspect in examining the plots, what's important for determining per-subscriber savings is the ratio of the incentive cost to the average monthly bill. The plots clearly show that for a wide range of assumptions concerning the average monthly bill, incentive cost, and retention rate, a significant cost savings is realized.

The plots assume that all subscribers identified by the predictor can be contacted and offered the incentive. If only some fraction F of all subscribers are contacted, then the estimated savings indicated by the plot should be multiplied by F.

To pin down a likely scenario, it is reasonable to assume that 50% of subscribers can be contacted, 35% of whom will be retained by offering an incentive that costs the carrier

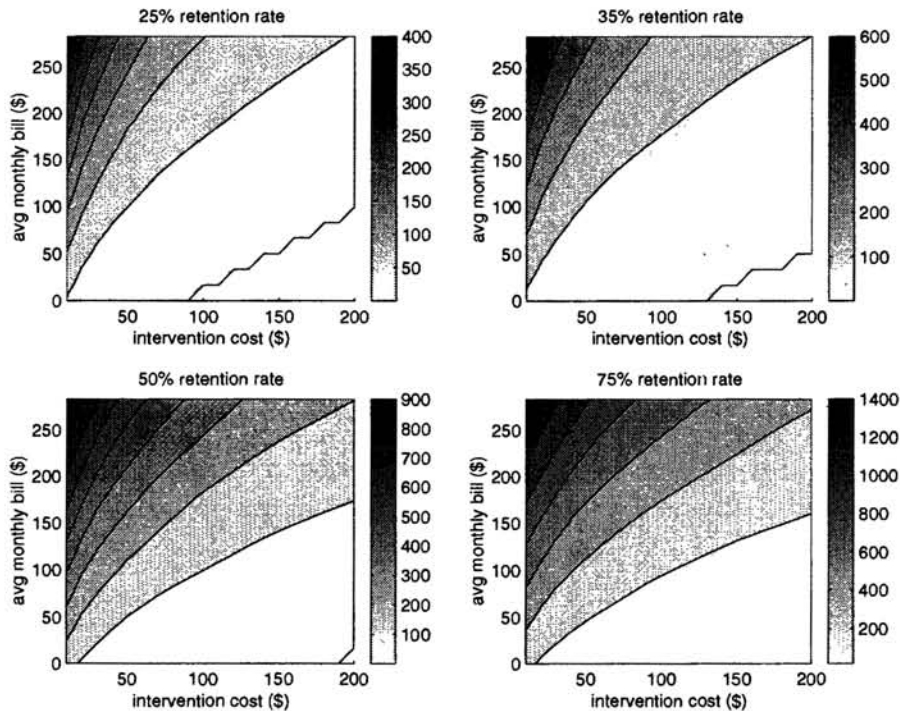

**FIGURE 3. Expected savings to the carrier per churnable subscriber, under a variety of assumptions concerning intervention cost, average monthly bill of subscriber, and retention rate that will be achieved by offering an incentive to a churnable subscriber.**

$75, and in our data base, the average monthly bill is $234. Under this scenario, the expected savings—above and beyond recovering the incentive cost—to the carrier is $93 based on the sophisticated neural net predictor. In contrast, the expected savings is only $47 based on the naive neural net predictor, and $81 based on the sophisticated logistic regression model. As we originally conjectured, both the nonlinearity of the neural net and the bias provided by the sophisticated representation are adding value to the predictions.

Our ongoing research involves extending these initial results in a several directions. First, we have confirmed our positive results with data from a different time window, and for test data from a later time window than the training data (as would be necessary in real-world usage). Second, we have further tuned and augmented our sophisticated representation to obtain higher prediction accuracy, and are now awaiting additional data to ensure the result replicates. Third, we are applying a variety of techniques, including sensitivity analysis and committee and boosting techniques, to further improve prediction accuracy. And fourth, we have begun to explore the consequences of iterating the decision making process and evaluating savings over an extended time period. Regardless of these directions for future work, the results presented here show the promise of data mining in the domain of wireless telecommunications. As is often the case for decision-making systems, the predictor need not be a perfect discriminator to realize significant savings.

# 6 REFERENCES

Bishop, C. (1995). Neural networks for pattern recognition. New York: Oxford University Press.

Fowlkes, A. J., Madan, A., Andrew, J., and Jensen, C.(1999). The effect of churn on value: An industry advisory.

Green, D. M., & Swets, J. A. (1966). *Signal detection theory and psychophysics.* New York: Wiley.

Luna, L. (1998). Churn is epidemic. Radio Communications Report, December 14, 1998.

Power, J. D., & Associates (1998). *1998 Residential Wireless Customer Satisfaction Survey.* September 22, 1998.
